# Reconstructing Patterns of Information Diffusion from Incomplete Observations [*]

**Flavio Chierichetti**
Department of Computer Science
Cornell University
Ithaca, NY 14853

**Jon Kleinberg**
Department of Computer Science
Cornell University
Ithaca, NY 14853

**David Liben-Nowell**
Department of Computer Science
Carleton College
Northfield, MN 55057

## Abstract

Motivated by the spread of on-line information in general and on-line petitions in particular, recent research has raised the following combinatorial estimation problem. There is a tree $T$ that we cannot observe directly (representing the structure along which the information has spread), and certain nodes randomly decide to make their copy of the information public. In the case of a petition, the list of names on each public copy of the petition also reveals a path leading back to the root of the tree. What can we conclude about the properties of the tree we observe from these revealed paths, and can we use the structure of the observed tree to estimate the size of the full unobserved tree $T$?

Here we provide the first algorithm for this size estimation task, together with provable guarantees on its performance. We also establish structural properties of the observed tree, providing the first rigorous explanation for some of the unusual structural phenomena present in the spread of real chain-letter petitions on the Internet.

## 1   Introduction

The on-line domain is a rich environment for observing *social contagion* — the tendency of new information, ideas, and behaviors to spread from person to person through a social network [1, 4, 6, 10, 12, 14, 17, 19]. When a link, invitation, petition, or other on-line item passes between people in the network, it is natural to model its spread using a tree structure: each person has the ability to pass the item to one or more others who haven't yet received it, producing a set of "offspring" in this tree. Recent work has considered such tree structures in the context of on-line conversations [13], chain letters [5, 9, 16], on-line product recommendations [11, 15], and other forms of forwarded e-mail [18]. These types of trees encode enormous detail about the process by which information spreads, but it has been a major methodological challenge to infer properties of their structure from the incomplete pictures of them that on-line data provides. Specifically, how do we reconstruct the paths followed by an on-line item, using our incomplete observations, and how do we estimate from these observations the total number of people who encountered the item?

A fundamental type of social contagion is one in which the item, by its very nature, accumulates information about the paths it follows as it travels through the social network. A canonical example

---

[*] A full version of this paper is available from the authors' Web pages.

of such a self-recording item is an on-line petition that spreads virally by e-mail — in other words, a *chain-letter petition* [9, 16]. Each recipient who wants to take part in the petition signs his or her name to it and forwards copies of it to friends. In this way, each copy of the petition contains a growing list of names that corresponds to a path all the way back to the initiator of the petition. Such types of petitions are a central ingredient in broader forms of Internet-based activism, a topic of considerable current interest [7, 8]. In the remainder of this discussion, we will refer to the item being spread as a "petition," although more generally we are considering any item with this self-recording structure.

**Reconstructing the Spread of Social Contagion.** Liben-Nowell and Kleinberg studied the following framework for reconstructing the spread of chain-letter petitions [16]. Empirical analyses of large-scale petitions suggest that the spreading pattern can be reasonably modeled as a tree $T$; although there are a small number of deviations, almost all participants sign a copy of the petition exactly once (even if they receive it multiple times), and so we can view the person from whom they received this copy as their parent in $T$. The originator of the petition is the root of $T$.

For a given petition, the tree $T$ is the structure we wish to understand, because it captures how the message spreads through the population. But in general we cannot hope to observe $T$: assuming that the petition spreads through individuals' e-mail accounts, hosted by multiple providers, there is no single organization that has all the information needed to reconstruct $T$.[1] Instead, we must obtain information about $T$ indirectly by a revelation mechanism that we can model as follows. For each person $v$ who signs the petition, there is a small probability $\delta > 0$ that $v$ will also publicly post their copy of it. In this case, we say that the node $v$ is *exposed*. When $v$ is exposed, we see not only that $v$ belongs to $T$, but we also see $v$'s path all the way back to the root $r$ of $T$, due to the list of names on $v$'s copy of the petition. Thus, if the set of people who post their copy of the petition is $\{v_1, v_2, \ldots, v_s\}$, then the subtree $T'$ of $T$ that we are able to observe consists precisely of the union of the $r$-to-$v_i$ paths in $T$ (for $i = 1, 2, \ldots, s$).[2]

We refer to this process as the $\delta$-*sampling* of a tree $T$: each node $v$ is exposed independently with probability $\delta$, and then all nodes on any path from the root to an exposed node (including the exposed nodes themselves) are *revealed*. This results in an observed tree, consisting of all revealed nodes, given by a random variable $T_\delta$ drawn from the set of all possible subtrees of $T$. Understanding the relationship between $T_\delta$ and $T$ is a fundamental question, since empirically we are often in a setting where we can observe $T_\delta$ and want to reason about properties of the larger unobserved tree $T$.

**Properties of $\delta$-Sampling: Some Basic Questions.** This is the basic issue we address in this paper: to understand the observation of a tree under $\delta$-sampling. In Liben-Nowell and Kleinberg's work, they looked at large trees revealed via the public posting of chain-letter petitions on the Internet — the real-life process that is mathematically abstracted by $\delta$-sampling — and they identified some unexpected and recurring empirical properties in the observed trees. In particular, the observed trees that they reconstructed had a very large *single-child fraction* — the fraction of nodes with only one child was above $94\%$. The resulting trees had a narrow, "stringy" appearance, owing to long chains of these single-child nodes; this led naturally to the question of why the patterns of chain-letter diffusion were giving rise to such structures. Possible answers were hypothesized in subsequent work. In particular, Golub and Jackson proposed an explanation based on computer simulation [9]; they studied a model for generating trees $T$ using a Galton–Watson branching process [3], and they showed that for branching processes near the critical value for extinction, $\delta$-sampling with small values of $\delta$ produced large single-child fractions in simulations.

This line of work has left open a number of questions, of which two principal questions are the following. First, can we provide a formal connection between $\delta$-sampling and the single-child fraction, and can we characterize the types of trees on which this connection holds (whether generated by branching processes or otherwise)? Second, existing work on this topic has so far not provided any framework capable of addressing what is perhaps the most basic question about $\delta$-sampling: given a tree $T_\delta$ with its set of exposed nodes indicated — i.e., a single outcome of the $\delta$-sampling process

— can we infer the number of nodes in the original tree $T$? (Note that we must do this inference without knowing the value of $\delta$.) This second question is a central issue in the sense that one generally asks, given partial observations of diffusion-based activism, for an estimate of the total number of people who were involved.

**Our Results: Single-Child Fractions and Size Estimation.** In this paper, we provide answers to both of these questions. First, we prove that $\delta$-sampling with small $\delta$ produces a large single-child fraction in all bounded-degree trees. We do not require any assumption that the unobserved tree $T$ arises from a branching process; the tree may be arbitrary as long as the degrees are bounded. More precisely, we show that for every natural number $k$, there is a function $f_k(x)$ for which $\lim_{x \to 0^+} f_k(x) = 0$, such that if $T$ has a maximum of $k$ children at any node, then $T_\delta$ has a single-child fraction of at least $1 - f_k(\delta)$ with high probability.[3] This result shows how the long stringy structures observed by Liben-Nowell and Kleinberg are a robust property of the process by which these structures were observed, essentially independently of what we assume (beyond a degree bound) for the structure of the unobserved tree.

Second, we consider the problem of estimating the *size* of $T$, which we define as the number of nodes it contains. In the basic formulation of the problem, we ask: given a single draw of $T_\delta$, with its set of exposed nodes indicated, can we estimate the size of $T$ to within a $1 \pm \varepsilon$ factor with high probability for any constant $\varepsilon > 0$? Here we show that this is possible for any bounded-degree tree, as well as for trees of unbounded degree that satisfy certain structural conditions.

Following our analysis of the estimation problem, we also consider the closely connected issue of *concentration*, which is related to estimation but distinct from it. Specifically, we ask: is it the case that the size of $T_\delta$ (a numerical random variable derived from $T_\delta$ itself) is concentrated near its mean? For sufficiently small $\delta$ the answer is no, and we give a bound on the threshold for $\delta$, tight to within an exponentially smaller term, at which concentration begins to hold. We note that concentration is a fundamentally different issue from estimation, in the sense that to be able to perform estimation, it is not sufficient that the size of $T_\delta$ be concentrated as a random variable.[4]

Using our methodology, we provide the first estimate for the reach of the Iraq-War protest chain letter studied by Liben-Nowell and Kleinberg: while the tree structure and rate of posting are at the limit of the parameters that can be handled, our framework estimates that their observed tree of 18,119 signers is a subtree of a larger unobserved tree with approximately 173,000 signers, which in total generated roughly 3.5 million copies of the e-mailed petition when both signers and non-signers are considered.

**Our Results: Extensions of the Basic Model.** Finally, we prove results for several extensions to our model. First, while we have focused on the case in which there is a fixed underlying tree $T$ which is then sampled using randomization, we can also define a model in which both $T$ and the sampling are the result of randomization — in particular, we consider a case in which $T$ is first generated from a critical Galton–Watson process [3], and then $\delta$-sampling is applied to the generated tree $T$. For this model, we show that as long as the offspring distribution of the Galton–Watson process has finite variance and unit expectation we can estimate the size of the unobserved tree $T$. Note that this allows for unbounded degrees — i.e., an offspring distribution with unbounded support — provided that the variance of this distribution is bounded.

A further extension relaxes the assumption that when a node $v$ makes its copy of the petition public, the path is revealed all the way back to the root. Instead of the full path being visible, one can alternately consider a situation in which only the previous $\ell$ names on the petition are preserved, and hence the observed tree can only be reconstructed if it is possible to piece these snippets of length $\ell$

together.[5] Here we can show that size estimation is possible provided that $\ell$ is at least $\delta^{-1}$ times a logarithmic factor, and this bound is asymptotically tight. Thus, our estimation methods work even when the data provides much less than a full path back to the root for each node made public. Due to space limitations, we defer details of this extension to the full version of the paper.

## 2   Single-Child Fraction

We begin by showing that in any bounded-degree tree, the fraction of single-child nodes converges to 1 as the sampling rate $\delta$ goes to 0. The plan for this proof is as follows. First of all, let the unobserved tree $T$ have $n$ nodes, each having at most $k$ children. Let us say that $v$ is a *branching node* if it has more than one child. (That is, we partition the nodes of the tree into three disjoint categories: leaves, single-child nodes, and branching nodes.) In any bounded-degree tree, the number of leaves and the number of branching nodes are within a constant factor of each other; in particular, this will be true in the revealed tree $T_\delta$.

Now, all leaves in $T_\delta$ are nodes that are exposed (i.e., made public) by the $\delta$-sampling process, so in expectation $T_\delta$ has at most $\delta n$ leaves (and we can bound the probability that the actual number exceeds this by more than a small factor). Thus, there will also be $O(\delta n)$ branching nodes, and all other nodes in $T_\delta$ must be single-child nodes.

Thus, the key to the argument is Theorem 2.1, which asserts that with high probability, $T_\delta$ has $\Omega(\delta n \log_k \delta^{-1})$ nodes in total. Since there are only $O(\delta n)$ leaves and branching nodes, the remaining nodes must be single-child nodes — and since the size of $T_\delta$ exceeds $O(\delta n)$ by a factor of $\Omega(\log_k \delta^{-1})$, the fraction of single-child nodes in $T_\delta$ must therefore converge to 1 as $\delta$ goes to 0.

Complete proofs of all the results in this paper are given in the full version; due to space limitations, we are not able to include them here. Where space permits, we will briefly summarize some of the proofs in the present version. For Theorem 2.1, the key is to show that in any bounded-degree tree $T$, we can identify $\Omega(\delta n)$ many disjoint sub-trees $T_1, T_2, T_3, \ldots$, each of size $\Theta(\delta^{-1})$. We then argue that in a constant fraction of these trees $T_i$, a node of $T_i$ at distance at least $\Omega(\log_k \delta^{-1})$ from $T_i$'s root will be exposed, which will result in the appearance of $\Omega(\log_k \delta^{-1})$ nodes in $T_i$.

**Theorem 2.1.** *Let $T$ be a rooted $n$-node tree, and suppose that no node in $T$ has more than $k \geq 2$ children.*[6]

*Let $\delta \leq k^{-\alpha}$, for any constant $\alpha > 2$. Let $T_\delta$ be the random subtree of $T$ revealed by the $\delta$-sampling process, and let $X_\delta$ be the number of its internal nodes. Then*

$$\Pr\left[X_\delta \geq \Omega(\delta n \log_k \delta^{-1})\right] \geq 1 - e^{-\Omega(\delta n)}.$$

We now follow the plan outlined at the beginning of this section, using this theorem to conclude that the fraction of single-child nodes converges to 1. Theorem 2.1 provided the main step; from here, we simply argue, in Theorem 2.2, that $T_\delta$ will have at most $O(\delta n)$ branching nodes with high probability.

**Theorem 2.2.** *Given a tree $T$ on $n$ nodes, a sampling rate $\delta$, and a number $M \leq n$, let $p$ be the probability that the size of the tree $T_\delta$ revealed by the $\delta$-sampling process is at most $M$. Let $m$ be the number of nodes in $T_\delta$ and $m_1$ be the number of single-child nodes in $T_\delta$. Then,*

$$\Pr\left[m_1 \geq \left(1 - O\left(\frac{\delta n}{M}\right)\right) \cdot m\right] \geq 1 - e^{-\Omega(\delta n)} - p.$$

Now, using Theorems 2.1 and 2.2, we obtain the main result about single-child nodes as the following corollary.

**Corollary 2.3.** *Let $T$ be a rooted $n$-node tree, and suppose that no node in $T$ has more than $k \geq 2$ children.*

*Let $\delta \leq k^{-\alpha}$, for any constant $\alpha > 1$. Let $T_\delta$ be the random subtree of $T$ revealed by the $\delta$-sampling process. Let $m$ and $m_1$ be, respectively, the number of nodes, and the number of nodes with exactly one child, in $T_\delta$. Then*

$$\Pr\left[m_1 \geq \left(1 - O\left(\frac{1}{\log_k \delta^{-1}}\right)\right) \cdot m\right] \geq 1 - e^{-\Omega(\delta n)}.$$

For concreteness, observe that if we choose $\delta = k^{-\Omega(1/\epsilon)}$ in Corollary 2.3, we obtain that the fraction of single-child nodes in the revealed tree will approach $1 - O(\epsilon)$ with probability $1 - \exp(-\Omega(\delta n))$.

## 3 Estimation

As before, given an unknown tree $T$, let $T_\delta$ be the tree revealed by the $\delta$-sampling process. In this section, we focus on the problem of size estimation: we present an algorithm which can be used as an unbiased estimator $\hat\delta$ for $\delta$, and then we estimate the size $n$ of the full unobserved tree.

Let $V = V(T_\delta)$ be the set of nodes of $T_\delta$, let $L \subseteq V$ be the set of its leaves, and let $E \subseteq V$ be the set of its nodes that were exposed. (Observe that $L \subseteq E$.) For the unbiased estimator $\hat\delta$, we consider the set of all nodes "above" the leaves of $T_\delta$ — that is, internal nodes on a path from a leaf of $T_\delta$ to the root — and we use the empirical fraction of exposures in this set as our value for $\hat\delta$.

After establishing that $\hat\delta$ is an unbiased estimator, we show that the probability of a large deviation between $\hat\delta$ and $\delta$ decreases exponentially in $|V - L|$, the number of internal nodes of $T_\delta$. Thus, to show a high probability bound for our size estimate, we need to establish a lower bound on the number of internal nodes of $T_\delta$, which will be the final step in the analysis.

We begin by describing an algorithm to produce the estimator $\hat\delta$, and a corresponding estimator $\hat n$ for the size of $T$.

- If $|V| = 0$ then return $\hat\delta = 0$; and if $|V| = 1$ then return $\hat\delta = 1$.
- Otherwise return $\hat\delta = \frac{|E| - |L|}{|V| - |L|}$. If $|E| > |L|$, also return $\hat n = \frac{|V| - |L|}{|E| - |L|} \cdot |E|$.

Observe that the algorithm is well-defined since, if $|V| \geq 2$, then $V$ will contain $T_\delta$'s root, which will not be contained in $L$, and therefore $|V| - |L| \geq 1$. For the following analysis of the algorithm observe that, since $L \subseteq E$ and $L \subseteq V$, we have $|E| - |L| = |E - L|$ and $|V| - |L| = |V - L|$.

We begin by showing that $\hat\delta$ is an unbiased estimator for $\delta$. Following the plan outlined above, we consider the independent exposures of all nodes that lie above the leaves of $T_\delta$, resulting in the set $E - L \subseteq V - L$. Because exposure decisions are made independently at each node, Chernoff bounds provide us with a concentration result.

**Lemma 3.1.** *$\hat\delta$ is an unbiased estimator for $\delta$. Furthermore, if $|V| \geq 2$,*

$$\Pr\left[\left|\hat\delta - \delta\right| \geq \epsilon \cdot \delta\right] \leq 2e^{-\frac{1}{3}\epsilon^2\delta|V-L|}.$$

We now transfer our bound on $|\hat\delta - \delta|$ to a bound on $|\hat n - n|$. For this, it suffices to combine three relationships among these quantities: (i) $\hat n = |E|/\hat\delta$ by definition; (ii) $|\hat\delta - \delta| \leq \epsilon \cdot \delta$ with high probability, by Lemma 3.1, and (iii) $||E| - \delta n| \leq \epsilon\delta n$ with high probability via Chernoff bounds, since the exposure decisions consist of $n$ independent coin flips each of probability $\delta$. Putting these together, we have the following corollary of Lemma 3.1.

**Corollary 3.2.** *If $|V| \geq 2$, then the size $n$ of the unknown tree $T$ satisfies*

$$\Pr\left[|n - \hat n| \leq \epsilon n\right] \geq 1 - e^{-\Theta(\epsilon^2\delta|V-L|)}.$$

### 3.1 Trees with sublinear maximum degree

Our bounds thus far show that $\hat n$ is close to $n$ with a probability that decreases exponentially in the number of internal nodes $|V - L|$. We now investigate cases under which we can replace this upper bound on the probability by a more powerful one that decreases exponentially in a function that depends directly on $n$.

To do this, we require a theorem that guarantees that the number of internal nodes is at least an explicit function of $n$; this function can then be used in place of $|V - L|$ in the probability bounds. Our main result for this purpose is the following; in many respects, the bound it establishes it is less refined than the bound from Theorem 2.1, but it is useful for obtaining a guarantee for the estimation procedure. The crux of the proof is to show that if a node $v$ has $k_v$ children, and $\delta k_v \leq 1$, then the probability that $v$ is revealed is at least a constant times the expected number of the exposed children of $v$: that is, $\Omega(\delta k_v)$; if, instead, $\delta k_v > 1$, then the probability that $v$ is revealed is $\Omega(1)$. The result then follows from a bound on the number of nodes of degree greater than $\delta^{-1}$, linearity of expectation, and Chernoff bounds.

**Theorem 3.3.** *Let $T$ be a rooted $n$-node tree, and suppose that no node in $T$ has more than $k \geq 1$ children.*

*Let $T_\delta$ be the random subtree of $T$ revealed by the $\delta$-sampling process. Then, the number $X_\delta$ of internal nodes of $T_\delta$ satisfies*

$$\Pr\left[X_\delta \geq \frac{1 - e^{-1}}{2} \cdot \min\left(k^{-1}, \delta\right) \cdot (n - 1)\right] \geq 1 - e^{-\Theta\left(n \, \min(k^{-1}, \delta)\right)}.$$

Using this theorem, we can directly replace the bound from Corollary 3.2 with one that is an explicit function of $n$. Specifically, the next result follows directly from Corollary 3.2 and Theorem 3.3.

**Corollary 3.4.** *Let $T$ be a rooted $n$-node tree, and suppose that no node in $T$ has more than $k \geq 1$ children. Then, the event*

$$(1 - \epsilon)n \leq \hat{n} \leq (1 + \epsilon)n$$

*happens with probability at least $1 - e^{-\Theta(\epsilon^2 \delta \min(\delta, k^{-1})n)}$.*

The smallest $\delta$ that Corollary 3.4 can tolerate is roughly $n^{-1/2}$: if $\delta \geq \Omega\left(\sqrt{\frac{\ln \eta^{-1}}{\epsilon^2 n}}\right)$, and no node in the unknown tree has more than $\delta^{-1}$ children (observe that $\delta^{-1} \gtrsim \sqrt{n}$), then with probability $1 - \eta$, the $\hat{n}$ returned by the estimator is within a multiplicative $1 \pm \epsilon$ factor of the actual $n$.

### 3.2 Trees Arising from Branching Processes

We observe that Corollary 3.2 can also be used, just as in Section 3.1, for critical branching processes — those whose offspring distributions have unit expectation. (We also require finite variance.) The main fact we require about such branching processes is that the height of a uniformly chosen node from a branching process tree (with offspring distribution having finite variance, unit expectation, and conditioned on being of size $n$) is at least $\Omega\left(n^{1/2-\epsilon}\right)$ with high probability [2].

Now, since $|V - L|$ is at least the length of the path joining a uniform chosen node to the root, if we choose $\delta \geq \Omega\left(n^{-1/2+2\epsilon}\right)$ it holds that $|V - L| \geq \omega(\delta^{-1})$, and Corollary 3.2 can be applied to obtain a concentration result for $\hat{n}$.

## 4 Concentration

As we observed in previous sections, the size of $T_\delta$ plays a prominent role in determining both the fraction of single-child nodes and the size of the unknown tree $T$.

In this section we prove some concentration results on the quantity $|T_\delta|$ — that is, we will bound the probability that $|T_\delta|$ is far from its mean, over random outcomes of the $\delta$-sampling process applied to the underlying tree $T$.

To begin with, the mean $E\left[|T_\delta|\right]$ depends not just on $|T|$ but also on the structure of $T$. However, it has a simple formulation in terms of this structure, as shown by the following claim, which is a direct application of linearity of expectation.

**Observation 4.1.** *Let $T$ be a rooted tree, and let $T_\delta$ be the random subtree of $T$ revealed by the $\delta$-sampling process. Then, if $|T_v|$ denotes the size of the subtree of $T$ rooted at $v$,*

$$E\left[|T_\delta|\right] = \sum_{v \in T} \left(1 - (1 - \delta)^{|T_v|}\right).$$

Our main result on concentration gives a value of $\delta$ above which $|T_\delta|$ has a high probability of being near its mean. The proof requires an intricate balancing of two kinds of nodes — those "high" in $T$, with many descendants, and those "low" in $T$, with few descendants. If there are many low nodes, then since their probabilities of being revealed behave relatively independently, we have concentration; if there only a few low nodes, then we have concentration simply from the fact that most of the high nodes will be revealed in almost all outcomes of the $\delta$-sampling process.

**Theorem 4.2.** *Let $T$ be a rooted tree on $n$ nodes, with height at most $H$. Let $T_\delta$ be the random subtree of $T$ revealed by the $\delta$-sampling process. Let $m$ be the size of $T_\delta$.*

*Then for any $\epsilon, \eta$ bounded above by some constant, and for any*

$$\delta \geq \Omega\left(\min\left(\frac{\ln^2 n/\eta}{\sqrt{n\epsilon^3\eta}}, \frac{H\ln^3 n/\eta}{n\epsilon^3\eta}\right)\right),$$

*it holds that $\Pr\left[|m - E[m]| \leq \epsilon E[m]\right] \geq 1 - \eta$.*

Note that the theorem requires a lower bound on the value of $\delta$, and we now show why this bound is necessary. In particular, we observe how the theorem does not hold if $\delta = o\left(n^{-1/2}\right)$. To do this, let $T$ be a tree whose root is connected directly to $n - 1 - \left\lceil\delta^{-1}\right\rceil$ leaves and also to a path of length $\left\lceil\delta^{-1}\right\rceil$. Then $T_\delta$ will not contain any node in the path with probability

$$(1-\delta)^{\left\lceil\delta^{-1}\right\rceil} \xrightarrow{\delta\to 0} e^{-1}.$$

If $T_\delta$ does not contain any node in the path, then it will only contain nodes adjacent to the root. Since there are $\Theta(n)$ of these nodes, it follows from Chernoff bounds that $T_\delta$ will contain at most $U = O(\delta n)$ many nodes.

On the other hand, the probability that $T_\delta$ will contain exactly one node in the path is

$$\left\lceil\delta^{-1}\right\rceil \cdot \delta \cdot (1-\delta)^{\left\lceil\delta^{-1}\right\rceil - 1} \xrightarrow{\delta\to 0} e^{-1}.$$

Since, under this conditioning, the single node in the path will be uniformly distributed over the path itself, with half the probability it will be in the lower half of the path — causing the upper half of the path to be revealed. Hence with constant probability at least $L = \Omega(\delta^{-1})$ nodes will be revealed.

We have shown that with constant probability the size of $T_\delta$ will be at most $U$, and with constant probability the size of $T_\delta$ will be at least $L$. If $\delta = o\left(n^{-1/2}\right)$, we have $L/U = \Omega(\delta^{-2}n^{-1}) = \omega(n)/n = \omega(1)$, from which it follows that the number of nodes of $T_\delta$ is not concentrated when $\delta = o\left(n^{-1/2}\right)$.

# 5   The Iraq-War Petition

Using the framework developed in the previous sections, we now turn to the anti-war petition studied by Liben-Nowell and Kleinberg. The petition, which protested the impending US-led invasion of Iraq, spread widely via e-mail in 2002–2003. The Iraq-War tree observed by Liben-Nowell and Kleinberg — after they did some mild preprocessing to clean the data — was deep and narrow, and contained the characteristic "stringy" pattern analyzed in Section 2, with over 94% of nodes having exactly one child. The observed Iraq-War tree contained $|V| = 18,119$ nodes and $|E| = 620$ exposed nodes, of which $|L| = 557$ were exposed leaves.

Using this information, we can apply the algorithm from Section 3: we estimate the posting probability as $\hat{\delta} = (620 - 557)/(18119 - 557) \approx 0.00359$, and we estimate the size of the unobserved Iraq-War tree to be $\hat{n} = |E|/\hat{\delta} \approx 172{,}832.38$ signatories.

We can also apply the results of Section 3 to analyze the error in our estimate $\hat{n}$. For this purpose, we pose the question concretely as follows: if the observed Iraq-War tree arose via $\delta$-sampling from an arbitrary unobserved tree $T$ of size $n$, what is the probability of the event that the estimate $\hat{n}$ produced by our algorithm lies in the interval $[\frac{1}{2}n, 2n]$? (Recall that our estimation algorithm is deterministic; the probability here is taken over the random choices of nodes exposed by the $\delta$-sampling process to the arbitrary fixed tree $T$.) We use a careful analysis (tight to constants), to show that the estimate $\hat{n}$ is quite tight, as indicated by the following theorem.

**Theorem 5.1.** *For any tree $T$ of size $n$, assuming the observed Iraq-War tree was produced via $\delta$-sampling of $T$, the event that $\hat{n}$ lies in the interval $[\frac{1}{2}n, 2n]$ is at least 95%.*

In addition to the number of signers of the petition, it is also of interest to determine the total number of e-mail messages generated by the spread of the petition. For this purpose, we first need to estimate the distribution of the number of recipients of an e-mailed copy of the petition.

To estimate this distribution, we collected a dataset of 147 copies of e-mail petitions with intact e-mail headers. In addition to data from the Iraq-War petition, these 147 copies include two other widely circulated petitions, supporting National Public Radio (NPR) and Mothers Against Drunk Driving (MADD). For each of these 147 e-mails, we counted the number of e-mail addresses to which the message was sent, including both direct and CCed recipients. (E-mails that were sent to mailing lists instead of to a list of individuals were not included in the set of 147.) The mean number of addressees was 20.37 (with standard deviation 20.60), the median was 14, and the maximum was 141. In addition to using the length of recipient lists to check the conditions needed for our theoretical results, we can also use these numbers to estimate the total reach of the Iraq-War petition. A person who signs the petition forwards the petition, on average, to 20.37 other addressees. Thus, by linearity of expectation, we can estimate that the $\approx 172,832.38$ signers in the unobserved tree sent a total of $\approx 3,520,595.58$ chain-letter e-mails in the Iraq-War petition.

Finally, the $\delta$-sampling process is a very simple abstraction of the process by which a widely circulated message becomes public, and with further inspection of the Iraq-War tree observed by Liben-Nowell and Kleinberg, we can begin to identify potential limitations of the basic $\delta$-sampling model. Principally, we have been assuming that each individual signatory of the petition exposes her petition copy independently with probability $\delta$. However, the assumption of independence of nodes' exposure events — while useful as an analytical abstraction — appears to be too simple to capture all the properties we see in the exposure events for the real data. One of the most common mechanisms that exposes a petition e-mail is when that e-mail is sent to a mailing list that archives its messages on the Web. When one person exposes her petition copy by sending it to a mailing list, then her friends are more likely to expose their petition copies by sending to the same list again, because they are more likely to be members of that same list (because of homophily) or because they "reply to all" (including the list) with their petition copy. We can quantify this independence issue explicitly by noting that many of the exposed internal nodes in the observed Iraq-War tree are close to the leaves of the tree. In particular, 48 of the 63 exposed internal nodes are within 10 hops of a leaf, out of only 5351 total such nodes. Thus the exposure rate for internal nodes within distance 10 of a leaf is $48/5351 \approx 0.00897$, while the exposure rate for internal nodes more than distance 10 from any leaf is $15/12211 \approx 0.00123$.

## 6 Conclusion

When information spreads through a social network, it often does so along a branching structure that can be reasonably modeled as a tree; but when we observe this spreading process, we frequently see only a portion of the full tree. In this work, we have developed techniques that allow us to reason about the full tree along which the information spreads from the portion that is observed; as a consequence, we are able to propose estimates for the size of a network cascade from a sample of it, and to deduce certain structural properties of the tree that it produces.

When we apply these techniques to data such as the Iraq-War petition in Section 5, our conclusions must clearly be interpreted in light of the model's underlying assumptions. Among these assumptions, the requirement of bounded degree may generally be fairly mild, since it essentially requires the tree of interest simply to be large enough compared to the number of children at any one node. Arguably more restrictive is the assumption that each node makes an independent decision about posting its copy of the information, and with the same fixed probability $\delta$. It is an interesting direction for further work to consider how one might perform comparable analyses with a relaxed version of these underlying assumptions, as well as the extent to which estimations of the type we have pursued here are robust in the face of different variations on the assumptions.

**Acknowledgements.** Supported in part by the MacArthur Foundation, a Google Research Grant, a Yahoo! Research Alliance Grant, and NSF grants IIS-0910664, CCF-0910940, and IIS-1016099.

## Footnotes

[1]Some petitions are hosted by a single Web site, rather than relying on social contagion; however, our focus here is on those that spread via person-to-person communication.

[2]In practice, there is a separate algorithmic question inherent in constructing this union in the presence of noise that makes different copies of the lists slightly typographically different from each other [5, 16], but this noise-correction process can be treated as a "black box" for our purposes here.

[3]Note that for simple reasons we need the given conditions on both $k$ and $\delta$. Indeed, if we don't bound the maximum number of children at any node, then the star graph — a single node with $n - 1$ children — has a single-child fraction of 0 with high probability for any non-trivial $\delta$. And if we don't consider the case of $\delta \to 0$, then each node with multiple children has a constant probability of having several of them made public and hence the single-child fraction can't converge to 1, unless the original tree was composed almost exclusively of single-child nodes to begin with.

[4]For example, if $T_\delta$ is simply a star with $s$ leaves, this observed tree is consistent with $\delta$-sampling applied to any $n$-leaf star, for any value of $n \geq s$ and with $\delta = s/n$.

[5]This version of the problem also arises naturally if we assume that individuals are not explicitly signing a petition, but that each forwarded message includes copies of the previous messages to a depth of $\ell$.

[6]If, in $T$, each node has at most one child, then $T$ is a path — in which case, an easy argument shows that almost every node will be revealed, and that necessarily only one of the revealed nodes will not have one child. Still, this case is covered by the theorem: just choose $k = 2$.

# References

[1] E. Adar, L. Zhang, L. A. Adamic, and R. M. Lukose. Implicit structure and the dynamics of blogspace. In *Workshop on the Weblogging Ecosystem*, 2004.

[2] D. Aldous. The continuum random tree II: An overview. In M. T. Barlow and N. H. Bingham, editors, *Stochastic Analysis*, pages 23–70. Cambridge University Press, 1991.

[3] K. B. Athreya and P. E. Ney. *Branching Processes*. Dover, 2004.

[4] E. Bakshy, B. Karrer, and L. A. Adamic. Social influence and the diffusion of user-created content. In *Proc. 10th ACM Conference on Electronic Commerce*, pages 325–334, 2009.

[5] C. H. Bennett, M. Li, and B. Ma. Chain letters and evolutionary histories. *Scientific American*, 288(6):76–79, June 2003.

[6] M. Cha, A. Mislove, and P. K. Gummadi. A measurement-driven analysis of information propagation in the flickr social network. In *Proc. 18th International World Wide Web Conference*, pages 721–730, 2009.

[7] J. Earl. The dynamics of protest-related diffusion on the web. *Information, Communication, and Society*, 13(26):209–225, 2010.

[8] R. K. Garrett. Protest in an information society: A review of literature on social movements and new ICTs. *Information, Communication, and Society*, 9(2):202–224, 2006.

[9] B. Golub and M. O. Jackson. Using selection bias to explain the observed structure of internet diffusions. *Proc. Natl. Acad. Sci. USA*, 107(24):10833–10836, 15 June 2010.

[10] D. Gruhl, R. V. Guha, D. Liben-Nowell, and A. Tomkins. Information diffusion through blogspace. In *Proc. 13th International World Wide Web Conference*, 2004.

[11] J. L. Iribarren and E. Moro. Impact of human activity patterns on the dynamics of information diffusion. *Physical Review Letters*, 103(3), July 2009.

[12] J. Kleinberg. Cascading behavior in networks: Algorithmic and economic issues. In N. Nisan, T. Roughgarden, É. Tardos, and V. Vazirani, editors, *Algorithmic Game Theory*, pages 613–632. Cambridge University Press, 2007.

[13] R. Kumar, M. Mahdian, and M. McGlohon. Dynamics of conversations. In *Proc. 16th ACM SIGKDD International Conference on Knowledge Discovery and Data Mining*, pages 553–562, 2010.

[14] J. Leskovec, L. Adamic, and B. Huberman. The dynamics of viral marketing. *ACM Transactions on the Web*, 1(1), May 2007.

[15] J. Leskovec, A. Singh, and J. M. Kleinberg. Patterns of influence in a recommendation network. In *Pacific-Asia Conference on Knowledge Discovery and Data Mining*, pages 380–389, 2006.

[16] D. Liben-Nowell and J. Kleinberg. Tracing information flow on a global scale using Internet chain-letter data. *Proc. Natl. Acad. Sci. USA*, 105(12):4633–4638, Mar. 2008.

[17] E. Sun, I. Rosenn, C. Marlow, and T. M. Lento. Gesundheit! Modeling contagion through Facebook News Feed. In *Proc. 3rd International Conference on Weblogs and Social Media*, 2009.

[18] D. Wang, Z. Wen, H. Tong, C.-Y. Lin, C. Song, and A.-L. Barabási. Information spreading in context. In *Proc. 20th International World Wide Web Conference*, pages 735–744, 2011.

[19] F. Wu, B. A. Huberman, L. A. Adamic, and J. R. Tyler. Information flow in social groups. *Physica A*, 337(1-2):327–335, 2004.

